# A METHOD FOR THE DESIGN OF STABLE LATERAL INHIBITION NETWORKS THAT IS ROBUST IN THE PRESENCE OF CIRCUIT PARASITICS

J.L. WYATT, Jr and D.L. STANDLEY
Department of Electrical Engineering and Computer Science
Massachusetts Institute of Technology
Cambridge, Massachusetts 02139

ABSTRACT

In the analog VLSI implementation of neural systems, it is sometimes convenient to build lateral inhibition networks by using a locally connected on-chip resistive grid. A serious problem of unwanted spontaneous oscillation often arises with these circuits and renders them unusable in practice. This paper reports a design approach that guarantees such a system will be stable, even though the values of designed elements and parasitic elements in the resistive grid may be unknown. The method is based on a rigorous, somewhat novel mathematical analysis using Tellegen's theorem and the idea of Popov multipliers from control theory. It is thoroughly practical because the criteria are local in the sense that no overall analysis of the interconnected system is required, empirical in the sense that they involve only measurable frequency response data on the individual cells, and robust in the sense that unmodelled parasitic resistances and capacitances in the inter-connection network cannot affect the analysis.

## I.  INTRODUCTION

The term "lateral inhibition" first arose in neurophysiology to describe a common form of neural circuitry in which the output of each neuron in some population is used to inhibit the response of each of its neighbors. Perhaps the best understood example is the horizontal cell layer in the vertebrate retina, in which lateral inhibition simultaneously enhances intensity edges and acts as an automatic gain control to extend the dynamic range of the retina as a whole[1]. The principle has been used in the design of artificial neural system algorithms by Kohonen[2] and others and in the electronic design of neural chips by Carver Mead et. al.[3,4].

In the VLSI implementation of neural systems, it is convenient to build lateral inhibition networks by using a locally connected on-chip resistive grid. Linear resistors fabricated in, e.g., polysilicon, yield a very compact realization, and nonlinear resistive grids, made from MOS transistors, have been found useful for image segmentation.[4,5] Networks of this type can be divided into two classes: feedback systems and feedforward-only systems. In the feedforward case one set of amplifiers imposes signal voltages or

currents on the grid and another set reads out the resulting response
for subsequent processing, while the same amplifiers both "write" to
the grid and "read" from it in a feedback arrangement. Feedforward
networks of this type are inherently stable, but feedback networks
need not be.

A practical example is one of Carver Mead's retina chips[3] that
achieves edge enhancement by means of lateral inhibition through a
resistive grid. Figure 1 shows a single cell in a continuous-time
version of this chip. Note that the capacitor voltage is affected
both by the local light intensity incident on that cell and by the
capacitor voltages on neighboring cells of identical design. Any
cell drives its neighbors, which drive both their distant neighbors
and the original cell in turn. Thus the necessary ingredients for
instability--active elements and signal feedback--are both present
in this system, and in fact the continuous-time version oscillates
so badly that the original design is scarcely usable in practice
with the lateral inhibition paths enabled.[6] Such oscillations can

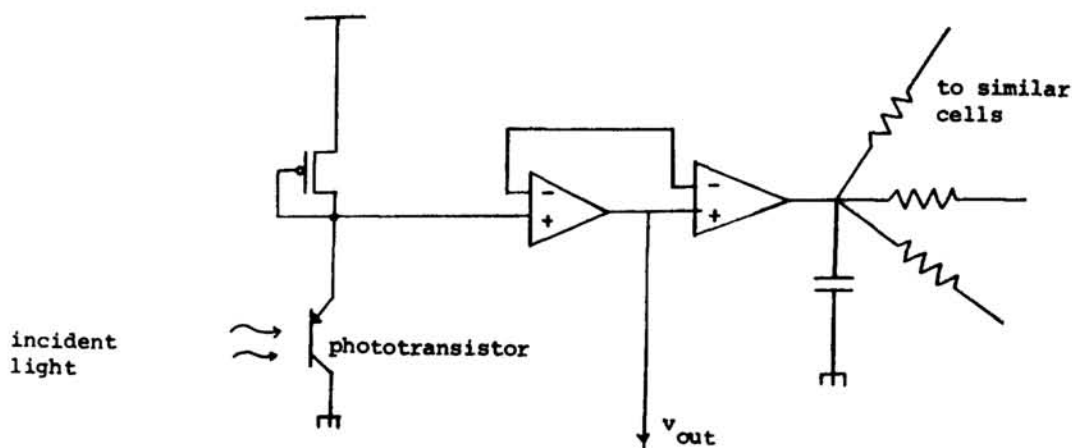

Figure 1. This photoreceptor and signal processor circuit, using two
MOS transconductance amplifiers, realizes lateral inhibition by
communicating with similar units through a resistive grid.

readily occur in any resistive grid circuit with active elements and
feedback, even when each individual cell is quite stable. Analysis
of the conditions of instability by straightforward methods appears
hopeless, since any repeated array contains many cells, each of
which influences many others directly or indirectly and is influenced
by them in turn, so that the number of simultaneously active feed-
back loops is enormous.

This paper reports a practical design approach that rigorously
guarantees such a system will be stable. The very simplest version
of the idea is intuitively obvious: design each individual cell so
that, although internally active, it acts like a passive system as
seen from the resistive grid. In circuit theory language, the
design goal here is that each cell's output impedance should be a
*positive-real*[7] function. This is sometimes not too difficult in
practice; we will show that the original network in Fig. 1 satisfies
this condition in the absence of certain parasitic elements. More
important, perhaps, it is a condition one can verify experimentally by frequency-response measurements.

It is physically apparent that a collection of cells that appear passive at their terminals will form a stable system when interconnected through a passive medium such as a resistive grid. The research contributions, reported here in summary form, are i) a demonstration that this passivity or positive-real condition is much stronger than we actually need and that weaker conditions, more easily achieved in practice, suffice to guarantee stability of the linear network model, and ii) an extension of i) to the *nonlinear* domain that furthermore rules out large-signal oscillations under certain conditions.

## II. FIRST-ORDER LINEAR ANALYSIS OF A SINGLE CELL

We begin with a linear analysis of an elementary model for the circuit in Fig. 1. For an initial approximation to the output admittance of the cell we simplify the topology (without loss of relevant information) and use a naive' model for the transconductance amplifiers, as shown in Fig. 2.

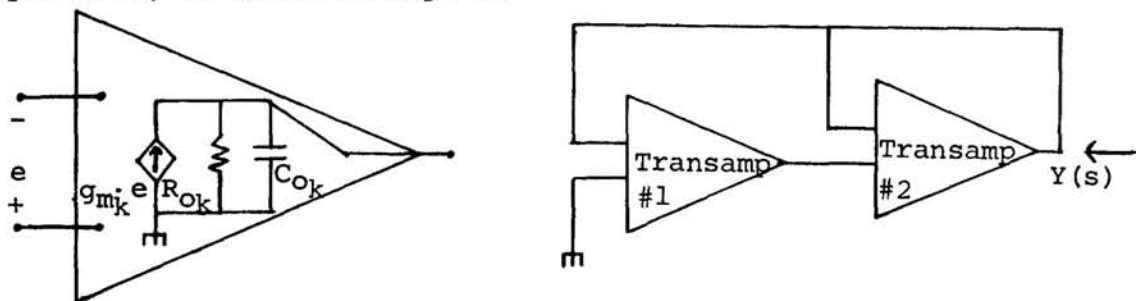

Figure 2. Simplified network topology and transconductance amplifier model for the circuit in Fig. 1. The capacitor in Fig. 1 has been absorbed into $C_{o_2}$.

Straightforward calculations show that the output admittance is given by

$$Y(s) = [g_{m_2} + R_{o_2}^{-1} + s\, C_{o_2}] + \frac{g_{m_1} g_{m_2} R_{o_1}}{(1 + s\, R_{o_1} C_{o_1})}. \tag{1}$$

This is a positive-real, i.e., passive, admittance since it can always be realized by a network of the form shown in Fig. 3, where $R_1 = (g_{m_2} + R_{o_2}^{-1})^{-1}$, $R_2 = (g_{m_1} g_{m_2} R_{o_1})^{-1}$, and $L = C_{o_1}/g_{m_1} g_{m_2}$.

Although the original circuit contains no inductors, the realization has both capacitors and inductors and thus is capable of damped oscillations. Nonetheless, *if* the transamp model in Fig. 2 were perfectly accurate, no network created by interconnecting such cells through a resistive grid (with parasitic capacitances) could exhibit sustained oscillations. For element values that may be typical in practice, the model in Fig. 3 has a lightly damped resonance around 1 KHz with a $Q \approx 10$. This disturbingly high Q suggests that the cell will be highly sensitive to parasitic elements not captured by the simple models in Fig. 2. Our preliminary

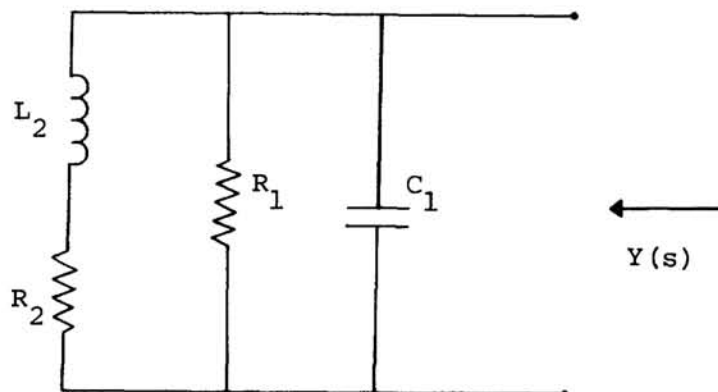

Figure 3. Passive network realization of the output admittance (eq. (1) of the circuit in Fig. 2.

analysis of a much more complex model extracted from a physical circuit layout created in Carver Mead's laboratory indicates that the output impedance will not be passive for all values of the trans-amp bias currents. But a definite explanation of the instability awaits a more careful circuit modelling effort and perhaps the design of an on-chip impedance measuring instrument.

### III. POSITIVE-REAL FUNCTIONS, θ-POSITIVE FUNCTIONS, AND STABILITY OF LINEAR NETWORK MODELS

In the following discussion $s = \sigma + j\omega$ is a complex variable, H(s) is a rational function (ratio of polynomials) in s with real coefficients, and we assume for simplicity that H(s) has no pure imaginary poles. The term *closed right half plane* refers to the set of complex numbers s with $\text{Re}\{s\} \geq 0$.

### Def. 1

The function H(s) is said to be *positive-real* if a) it has no poles in the right half plane and b) $\text{Re}\{H(j\omega)\} \geq 0$ for all $\omega$.

If we know at the outset that H(s) has no right half plane poles, then Def. 1 reduces to a simple graphical criterion: H(s) is positive-real if and only if the *Nyquist diagram* of H(s) (i.e. the plot of $H(j\omega)$ for $\omega \geq 0$, as in Fig. 4) lies entirely in the closed right half plane.

Note that positive-real functions are necessarily stable since they have no right half plane poles, but stable functions are not necessarily positive-real, as Example 1 will show.

A deep link between positive real functions, physical networks and passivity is established by the classical result[7] in linear circuit theory which states that H(s) is positive-real if and only if it is possible to synthesize a 2-terminal network of positive linear resistors, capacitors, inductors and ideal transformers that has H(s) as its driving-point impedance or admittance.

## Def. 2

The function H(s) is said to be θ-positive for a particular value of θ(θ ≠ 0, θ ≠ π), if a) H(s) has no poles in the right half plane, and b) the Nyquist plot of H(s) lies strictly to the right of the straight line passing through the origin at an angle θ to the real positive axis.

Note that every θ-positive function is stable and any function that is θ-positive with θ = π/2 is necessarily positive-real.

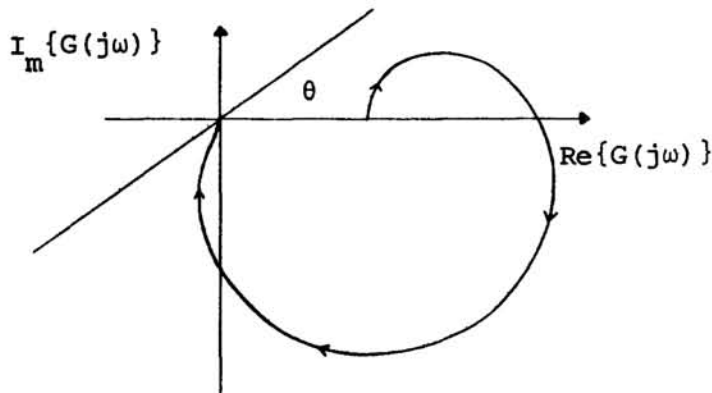

Figure 4. Nyquist diagram for a function that is θ-positive but not positive-real.

## Example 1

The function

$$G(s) = \frac{(s+1)(s+40)}{(s+5)(s+6)(s+7)} \tag{2}$$

is θ-positive (for any θ between about 18° and 68°) and stable, but it is not positive-real since its Nyquist diagram, shown in Fig. 4, crosses into the left half plane.

The importance of θ-positive functions lies in the following observations: 1) an interconnection of passive linear resistors and capacitors and cells with stable linear impedances can result in an unstable network, b) such an instability cannot result if the impedances are also positive-real, c) θ-positive impedances form a larger class than positive-real ones and hence θ-positivity is a less demanding synthesis goal, and d) Theorem 1 below shows that such an instability cannot result if the impedances are θ-positive, even if they are not positive-real.

## Theorem 1

Consider a linear network of arbitrary topology, consisting of any number of passive 2-terminal resistors and capacitors of arbitrary value driven by any number of active cells. If the output impedances

of all the active cells are $\theta$-positive for some common $\theta$, $0<\theta\leq\frac{\pi}{2}$, then the network is stable.

The proof of Theorem 1 relies on Lemma 1 below.

## Lemma 1

If $H(s)$ is $\theta$-positive for some fixed $\theta$, then for all $s_o$ in the closed first quadrant of the complex plane, $H(s_o)$ lies strictly to the right of the straight line passing through the origin at an angle $\theta$ to the real positive axis, i.e., $Re\{s_o\} \geq 0$ and $Im\{s_o\} \geq 0 \Rightarrow$ $\theta-\pi < \angle H(s_o) < \theta$.

## Proof of Lemma 1 (Outline)

Let $d$ be the function that assigns to each $s$ in the closed right half plane the perpendicular distance $d(s)$ from $H(s)$ to the line defined in Def. 2. Note that $d(s)$ is harmonic in the closed right half plane, since $H$ is analytic there. It then follows, by application of the maximum modulus principle[8] for harmonic functions, that $d$ takes its minimum value on the boundary of its domain, which is the imaginary axis. This establishes Lemma 1.

## Proof of Theorem 1 (Outline)

The network is unstable or marginally stable if and only if it has a natural frequency in the closed right half plane, and $s_o$ is a natural frequency if and only if the network equations have a nonzero solution at $s_o$. Let $\{I_k\}$ denote the complex branch currents of such a solution. By Tellegen's theorem[9] the sum of the complex powers absorbed by the circuit elements must vanish at such a solution, i.e.,

$$\sum_{\substack{resistances}} R_k|I_k|^2 + \sum_{\substack{capacitances}} |I_k|^2/s_o C_k + \sum_{\substack{cell \\ terminal\ pairs}} Z_k(s_o)|I_k|^2 = 0,$$

(3)

where the second term is deleted in the special case $s_o=0$, since the complex power into capacitors vanishes at $s_o=0$.

If the network has a natural frequency in the closed right half plane, it must have one in the closed first quadrant since natural frequencies are either real or else occur in complex conjugate pairs. But (3) cannot be satisfied for any $s_o$ in the closed first quadrant, as we can see by dividing both sides of (3) by $\sum_k |I_k|^2$, where the

sum is taken over all network branches. After this division, (3) asserts that zero is a convex combination of terms of the form $R_k$, terms of the form $(C_k s_o)^{-1}$, and terms of the form $Z_k(s_o)$. Visualize where these terms lie in the complex plane: the first set lies on the real positive axis, the second set lies in the closed 4-th quadrant since $s_o$ lies in the closed 1st quadrant by assumption, and the third set lies to the right of a line passing through the origin at an angle $\theta$ by Lemma 1. Thus all these terms lie strictly to the right of this line, which implies that no convex combination of them can equal zero. Hence the network is stable!

## IV.  STABILITY RESULT FOR NETWORKS WITH NONLINEAR RESISTORS AND CAPACITORS

The previous result for linear networks can afford some limited insight into the behavior of nonlinear networks.  First the nonlinear equations are linearized about an equilibrium point and Theorem 1 is applied to the linear model.  If the linearized model is stable, then the equilibrium point of the original nonlinear network is *locally stable*, i.e., the network will return to that equilibrium point *if* the initial condition is sufficiently near it.  But the result in this section, in contrast, applies to the full nonlinear circuit model and allows one to conclude that in certain circumstances the network cannot oscillate *even if* the initial state is *arbitrarily far from* the equilibrium point.

### Def. 3

A function $H(s)$ as described in Section III is said to satisfy the *Popov criterion*[10] if there exists a real number $r > 0$ such that $\text{Re}\{(1+j\omega r)\, H(j\omega)\} \geq 0$ for all $\omega$.

Note that positive real functions satisfy the Popov criterion with $r = 0$.  And the reader can easily verify that $G(s)$ in Example 1 satisfies the Popov criterion for a range of values of $r$. The important effect of the term $(1+j\omega r)$ in Def. 3 is to rotate the Nyquist plot counterclockwise by progressively greater amounts up to 90° as $\omega$ increases.

### Theorem 2

Consider a network consisting of nonlinear 2-terminal resistors and capacitors, and cells with linear output impedances $Z_k(s)$.  Suppose

i) the resistor curves are characterized by continuously differentiable functions $i_k = g_k(v_k)$ where $g_k(0) = 0$ and $0 < g_k'(v_k) < G < \infty$ for all values of $k$ and $v_k$,

ii) the capacitors are characterized by $i_k = C_k(v_k)\dot{v}_k$ with $0 < C_1 < C_k(v_k) < C_2 < \infty$ for all values of $k$ and $\dot{v}_k$,

iii) the impedances $Z_k(s)$ have no poles in the closed right half plane and all satisfy the Popov criterion for some common value of $r$.

If these conditions are satisfied, then the network is stable in the sense that, for any initial condition,

$$\int_0^{\infty}\left[\sum_{\text{all branches}} i_k^2(t)\right] dt < \infty . \tag{4}$$

The proof, based on Tellegen's theorem, is rather involved.  It will be omitted here and will appear elsewhere.

ACKNOWLEDGEMENT

    We sincerely thank Professor Carver Mead of Cal Tech for enthusiastically supporting this work and for making it possible for us to present an early report on it in this conference proceedings. This work was supported by Defense Advanced Research Projects Agency (DoD), through the Office of Naval Research under ARPA Order No. 3872, Contract No. N00014-80-C-0622 and Defense Advanced Research Projects Agency (DARPA) Contract No. N00014-87-K-0825.

REFERENCES

1.  F.S. Werblin, "The Control of Sensitivity on the Retina," Scientific American, Vol. 228, no. 1, Jan. 1983, pp. 70-79.
2.  T. Kohonen, Self-Organization and Associative Memory, (vol. 8 in the Springer Series in Information Sciences), Springer Verlag, New York, 1984.
3.  M.A. Sivilotti, M.A. Mahowald, and C.A. Mead, "Real Time Visual Computations Using Analog CMOS Processing Arrays," Advanced Research in VLSI - Proceedings of the 1987 Stanford Conference, P. Losleben, ed., MIT Press, 1987, pp. 295-312.
4.  C.A. Mead, Analog VLSI and Neural Systems, Addison-Wesley, to appear in 1988.
5.  J. Hutchinson, C. Koch, J. Luo and C. Mead, "Computing Motion Using Analog and Binary Resistive Networks," submitted to IEEE Transactions on Computers, August 1987.
6.  M. Mahowald, personal communication.
7.  B.D.O. Anderson and S. Vongpanitlerd, Network Analysis and Synthesis - A Modern Systems Theory Approach, Prentice-Hall, Englewood Cliffs, NJ., 1973.
8.  L.V. Ahlfors, Complex Analysis, McGraw-Hill, New York, 1966, p. 164.
9.  P. Penfield, Jr., R. Spence, and S. Duinker, Tellegen's Theorem and Electrical Networks, MIT Press, Cambridge, MA, 1970.
10. M. Vidyasagar, Nonlinear Systems Analysis, Prentice-Hall, Englewood Cliffs, NJ, 1970, pp. 211-217.

